# Saliency-Driven Image Acuity Modulation on a Reconfigurable Silicon Array of Spiking Neurons

**R. Jacob Vogelstein**[1], **Udayan Mallik**[2], **Eugenio Culurciello**[3],
**Gert Cauwenberghs**[2] **and Ralph Etienne-Cummings**[2]

[1]Dept. of Biomedical Engineering, Johns Hopkins University, Baltimore, MD
[2]Dept. of Electrical & Computer Engineering, Johns Hopkins University, Baltimore, MD
[3]Dept. of Electrical Engineering, Yale University, New Haven, CT
{*jvogelst,umallik1,gert,retienne*}*@jhu.edu, eugenio.culurciello@yale.edu*

## Abstract

We have constructed a system that uses an array of 9,600 spiking silicon neurons, a fast microcontroller, and digital memory, to implement a reconfigurable network of integrate-and-fire neurons. The system is designed for rapid prototyping of spiking neural networks that require high-throughput communication with external address-event hardware. Arbitrary network topologies can be implemented by selectively routing address-events to specific internal or external targets according to a memory-based projective field mapping. The utility and versatility of the system is demonstrated by configuring it as a three-stage network that accepts input from an address-event imager, detects salient regions of the image, and performs spatial acuity modulation around a high-resolution *fovea* that is centered on the location of highest salience.

## 1 Introduction

The goal of neuromorphic engineering is to design large-scale sensory information processing systems that emulate the brain. In many biological neural systems, the information received by a sensory organ passes through multiple stages of neural computations before a judgment is made. A convenient way to study this functionality is to design separate chips for each stage of processing and connect them with a fast data bus. However, it is not always advisable to fabricate a new chip to test a hypothesis regarding a particular neural computation, and software models of spiking neural networks cannot typically execute or communicate with external devices in real-time. Therefore, we have designed specialized hardware that implements a reconfigurable array of spiking neurons for rapid prototyping of large-scale neural networks.

Neuromorphic sensors can generate up to millions of spikes per second (see, e.g., [1]), so a proper communication protocol is required for multi-chip systems. "Address-Event Representation" (AER) was developed for this purpose over a decade ago and has since become the common "language" of neuromorphic chips [2–7]. The central idea of AER is to use time-multiplexing to emulate extensive connectivity between neurons. Although it was originally proposed to implement a one-to-one connection topology, AER has been extended to allow convergent and divergent connectivity [5, 8, 9], and has even been used

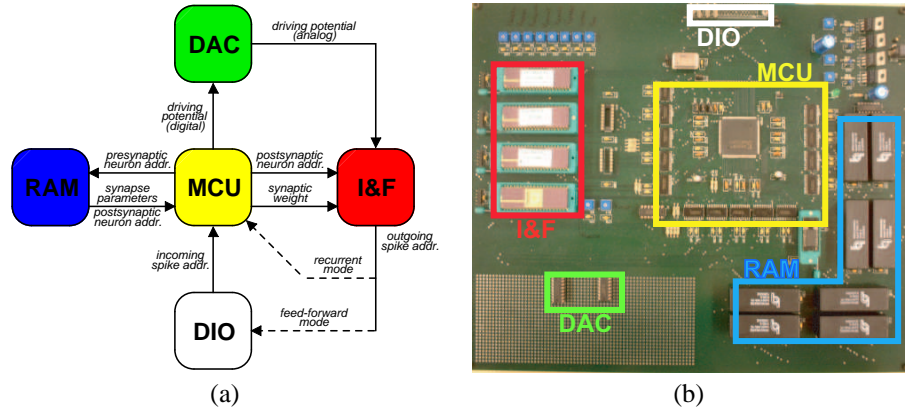

<div style="text-align:center">(a)         (b)</div>

Figure 1: (a) Block diagram of IFAT system. Incoming and outgoing address-events are communicated through the digital I/O port (DIO), with handshaking executed by the microcontroller (MCU). The digital-to-analog converter (DAC) is controlled by the MCU and provides the synaptic driving potential ('E' in Figure 2) to the integrate-and-fire neurons (I&F), according to the synapse parameters stored in memory (RAM). Modified from [18]. (b) Printed circuit board integrating all components of the IFAT system.

for functions in addition to inter-chip communication [10–12]. Within our hardware array, all inter-neuron communication is performed using AER; the absence of hardwired connections is the feature that allows for reconfigurability.

A few examples of AER-based reconfigurable neural array transceivers can be found in the literature [8, 9], but our Integrate-and-Fire Array Transceiver system (IFAT) differs in its size and flexibility. With four custom aVLSI chips [13] operating in parallel and 128 MB of digital RAM, the system contains 9,600 neurons and up to 4,194,304 synapses. Because it was designed from the start for generality and biological realism, every silicon neuron implements a discrete-time version of the classical biological "membrane equation" [14], a simple conductance-like model of neural function that allows for emulating an unlimited number of synapse types by dynamically varying two parameters [13]. By using a memory-based projective field mapping to route incoming address-events to different target neurons, the system can implement arbitrarily complex network topologies, limited only by the capacity of the RAM.

To demonstrate the functionality of the IFAT, we designed a three-stage feed-forward model of salience-based attention and implemented it entirely on the reconfigurable array. The model is based on a biologically-plausible architecture that has been used to explain human visual search strategies [15, 16]. Unlike previous hardware implementations (e.g. [17]), we use a multi-chip system and perform all computations with spiking neurons. The network accepts spikes from an address-event imager as inputs, computes spatial derivatives of light intensity as a measure of local information content, identifies regions of high salience, and foveates a location of interest by reducing the resolution in the surrounding areas. These capabilities are useful for smart, low-bandwidth, wide-angle surveillance networks.

## 2 Hardware

From the perspective of an external device, the IFAT system (Figure 1) operates as an AER transceiver, both receiving and transmitting spikes over a bidirectional address-event (AE) bus. Internally, incoming events are routed to any number of integrate-and-fire (I&F)

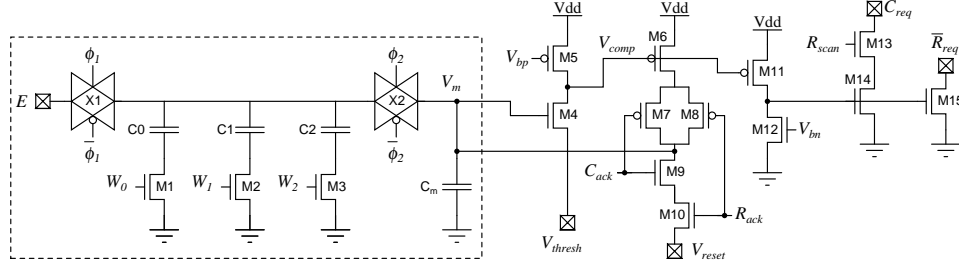

Figure 2: Silicon neuron. The "general-purpose" synapse is shown inside the dashed box [13], with event generation circuitry shown right [9].

neurons according to a look-up table stored in RAM. When the inputs are sufficient to cause a neuron to spike, the output is either directed to other internal neurons (for recurrent networks) or to an external device via the AE bus. The following two sections will describe the system and silicon neurons in more detail.

## 2.1 The IFAT system

A block diagram of the IFAT system and its physical implementation are shown in Figure 1. The primary components are a 100 MHz FPGA (Xilinx XC2S100PQ208), 128 MB of non-volatile SRAM (TI bq4017), a high-speed DAC (TI TLC7524CN), a 68-pin digital I/O interface (DIO), and 4 custom aVLSI chips that implement a total of 9,600 I&F neurons. The FPGA controls access to both an internal and external AE bus, and communicates address-events between both the I&F neurons and external devices in bit-parallel using a four-phase asynchronous handshaking scheme.

The 128 MB of RAM is arranged as a 4 MB $\times$ 32 array. Each 32-bit entry contains a complete description of a single synapse, specifying the postsynaptic target, the synaptic equilibrium potential, and the synaptic weight. The weight field can be further subdivided into three parts, corresponding to three ways in which biological neurons can control the synaptic weight ($w$) [14, p. 91]:

$$w = npq \tag{1}$$

where $n$ is the number of quantal neurotransmitter sites, $p$ is the probability of neurotransmitter release per site, and $q$ is a measure of the postsynaptic effect of the neurotransmitter. In the IFAT system, the FPGA can implement $p$ with a simple pseudo-random number algorithm, it can control $n$ by sending multiple outgoing spikes for each incoming spike, and it sends the value of $q$ to the I&F neuron chips (see Section 2.2).

Instead of hardwired connections between neurons, the IFAT implements "virtual connections" by serially routing incoming events to their appropriate targets at a rate of up to 1,000,000 events per second. When the IFAT receives an AE from an external device, the FPGA observes the address, appends some "chip identifier" (CID) bits, and stores the resulting binary number as a `base address`. It then adds additional `offset` bits to form a complete 22-bit RAM address, which it uses to look up a set of synaptic parameters. After configuring $q$ and instructing the DAC to produce the analog synaptic equilibrium potential, the FPGA activates a target neuron by placing its address on the internal AE bus and initiating asynchronous handshaking with the appropriate I&F chip. It then increments the `offset` by one and repeats the process for the next synapse, stopping when it sees a reserved code word in the data field. Recurrent connections can be implemented simply by appending a different CID to events generated by the on-board I&F neurons, while connections to external devices are achieved by specifying the appropriate CID for

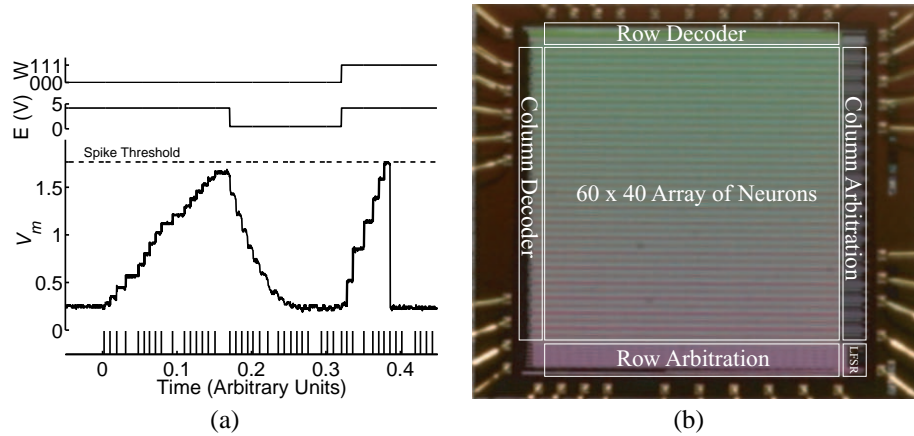

(a)                    (b)

Figure 3: (a) Data collected from one neuron during operation of the chip. The lower trace illustrates the membrane potential ($V_m$) of a single neuron in the array as a series of events are sent at times marked at the bottom of the figure. The synaptic equilibrium potential ($E$) and synaptic weight ($W$) are drawn in the top two traces. Figure from [13]. (b) Integrate-and-fire chip micrograph. The linear-feedback shift register (LFSR) implements a pseudo-random element for resolving arbitration conflicts. Modified from [13].

the postsynaptic target. With this infrastructure, arbitrary patterns of connectivity can be implemented, limited only by the memory's capacity.

## 2.2 Integrate-and-Fire Neurons

As described above, the IFAT system includes four custom aVLSI chips [13] that contain a total of 9,600 integrate-and-fire neurons. All the neurons are identical and each implements a simple conductance-like model of a single, "general purpose" synapse using a switched-capacitor architecture (Figure 2). The synapses have two internal parameters that can be dynamically modulated for each incoming event: the synaptic equilibrium potential ($E$) and the synaptic weight (W0-W2). Values for both parameters are stored in RAM; the 3-bit $q$ is used by the FPGA to selectively enable binary-sized capacitors C0-C2, while $E$ is converted to an analog value by the DAC. By varying these parameters, it is possible to emulate a large number of different kinds of synapses impinging on the same cell. An example of one neuron in operation is shown in Figure 3a.

A micrograph of the integrate-and-fire chip is shown in Figure 3b. Incoming address-events are decoded and sent to the appropriate neuron in the 60 × 40 array. When a neuron's membrane potential exceeds an externally-provided threshold voltage, it requests service from the peripheral arbitration circuitry. After request is acknowledged, the neuron is reset and its address is placed on the IFAT system's internal AER bus. Conflicts between simultaneously active neurons are resolved by a novel arbitration scheme that includes a pseudo-random element on-chip [19].

## 3 Experimental Design and Results

To demonstrate the functionality of the IFAT system, we designed and implemented a three-stage network for salience-based foveation [16] of an address-event imager. This work is motivated by the fact that wide-angle image sensors in a monitoring sensor network

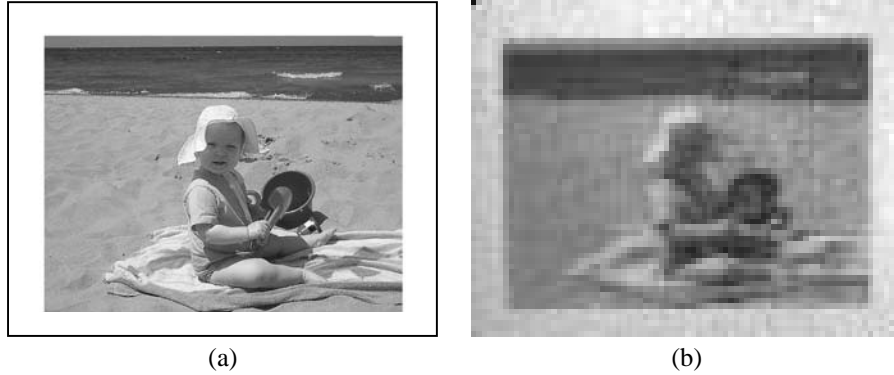

Figure 4: (a) Test image. (b) Output from Octopus Retina.

extract a large quantity of data from the environment, most of which is irrelevant. Because bandwidth is limited and data transmission is energy-intensive, it is desirable to reduce the amount of information sent over the communication channel. Therefore, if a particular region of the visual field can be identified as having high salience, that part of the image can be selectively transmitted with high resolution and the surrounding scene can be be compressed.

The input to the first stage of the network is a stream of address-events generated by an asynchronous imager called the "Octopus Retina" (OR) [1,20]. The OR contains a $60 \times 80$ array of light-sensitive "neurons" that each represent local light intensity as a spike rate. In other words, pixels that receive a lot of light spike more frequently than those that receive a little light. For these experiments, we collected 100,000 events from the OR over the course of about one second while it was viewing a grayscale picture mounted on a white background. The test image and OR output are shown in Figure 4.

To identify candidate regions of salience, the first stage of the network is configured to compute local changes in contrast. Every $2 \times 2$ block of pixels in the OR corresponds to four neurons on the IFAT that respond to light-to-dark or dark-to-light transitions in the rightward or downward direction (Figure 5a). Each IFAT cell computes local changes in contrast due to a receptive field (RF) that spans four OR pixels in either the horizontal or vertical dimension, with two of its inputs being excitatory and the other two being inhibitory. When a given IFAT cell's RF falls on a region of visual space with uniform brightness, all of the OR pixels projecting to that cell will have the same mean firing rate, so the excitatory and inhibitory inputs will cancel. However, if a cell's excitatory inputs are exposed to high light intensity and its inhibitory inputs are exposed to low light intensity, the cell will receive more excitatory inputs than inhibitory inputs and will generate an output spike train with spike frequency proportional to the contrast. The output from the 4,800 IFAT neurons in the first stage of the network in response to the OR input is shown in Figure 5b.

The second stage of processing is designed to pool inputs from neighboring contrast-sensitive cells to identify locations of high salience. Our underlying assumption is that regions of interest will contain more detail than their surroundings, producing a large output from the first stage. Blocks of $8 \times 8$ IFAT cells from the first stage project to single cells in the second stage, and each $8 \times 8$ region overlaps the next by 4 neurons (Figure 6a). Therefore, every IFAT cell in the second stage has an $8 \times 8$ RF. Although it is not necessary to normalize the firing rates of the first and second stages, because every second stage IFAT cell receives 64 inputs, we reduce the strength of the synaptic connections between the two stages to conserve bandwidth. The output from the 300 IFAT neurons in the second

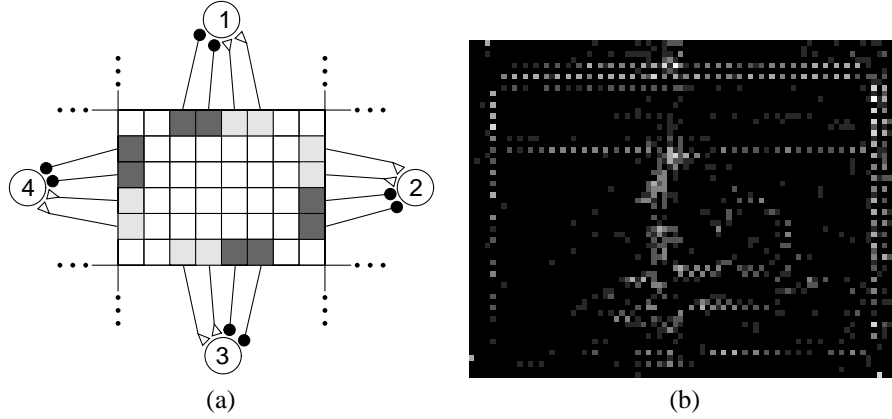

(a)                                          (b)

Figure 5: (a) Stage 1 network for computing local changes in contrast. Squares in the center represent OR pixels. Circles represent IFAT neurons. Excitatory synapses are represented by triangles, and inhibitory synapses as circles. Only four IFAT neurons with non-overlapping receptive fields are shown for clarity. (b) Output of stage 1, as implemented on the IFAT, with Figure 4b from the OR as input.

stage of the network in response to the output from the first stage IFAT neurons is shown in Figure 6b.

The final stage of processing modulates the spatial acuity of the original image to reduce the resolution outside the region of highest salience. This is achieved by a foveation network that pools inputs from neighboring pixels using overlapping Gaussian kernels (Figure 7a) [18]. The shape of the kernel functions is implemented by varying the synaptic weight and synaptic equilibrium potential between OR neurons and IFAT cells in the third stage: within every pooled block, the strongest connections originate from the center pixels and the weakest connections come from the outermost pixels. Instead of physically moving the OR imager to center the fovea on the region of interest, we relocate the fovea by performing simple manipulations in the address domain. First, the address space of incoming events is enlarged beyond the range provided by the OR and the fovea is centered within this virtual visual field (Figure 7a). Then, the row and column address of the second stage IFAT neuron with the largest output is subtracted from the address of the center of the fovea, and the result is stored as a constant offset. This offset is then added to the addresses of all incoming events from the OR, resulting in a shift of the OR image in the virtual visual field so that the fovea will be positioned over the region of highest salience. The output from the 1,650 IFAT neurons in the third stage network is shown in Figure 7b. With a $32 \times 32$ pixel high-resolution fovea, the network allows for a 66% reduction in the number of address-events required to reconstruct the image.

## 4   Conclusion

We have demonstrated a multi-chip neuromorphic system for performing saliency-based spatial acuity modulation. An asynchronous imager provides the input and communicates with a reconfigurable array of spiking silicon neurons using address-events. The resulting output is useful for efficient spatial and temporal bandwidth allocation in low-power vision sensors for wide-angle video surveillance. Future work will concentrate on extending the functionality of the multi-chip system to perform stereo processing on address-event data from two imagers.

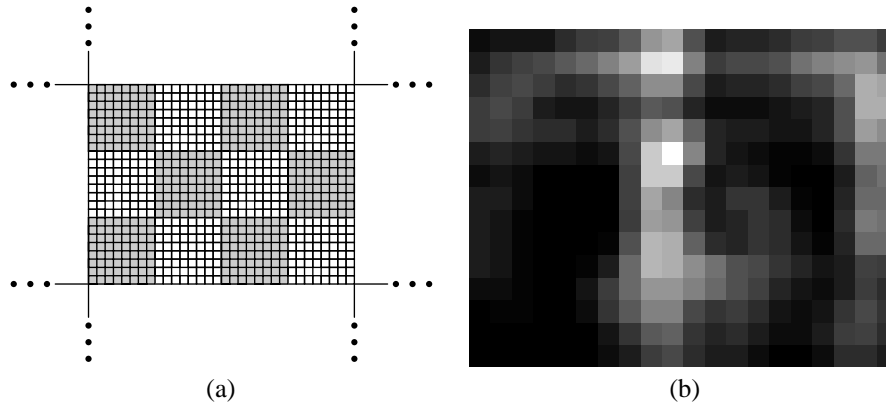

(a)  (b)

Figure 6: (a) Stage 2 network for computing local changes in contrast. Blocks of $8 \times 8$ IFAT neurons from stage 1 (shown as regions alternately shaded white and gray) project to single IFAT neurons in stage 2 (not shown). Blocks are shown as non-overlapping for clarity. (b) Output of stage 2, as implemented on the IFAT, with Figure 5b from stage 1 as input.

### Acknowledgments

This work was partially funded by NSF Awards #0120369, #9896362, and IIS-0209289; ONR Award #N00014-99-1-0612; and a DARPA/ONR MURI #N00014-95-1-0409. Additionally, RJV is supported by an NSF Graduate Research Fellowship.

## References

[1] E. Culurciello, R. Etienne-Cummings, and K. A. Boahen, "A biomorphic digital image sensor," *IEEE J. Solid-State Circuits*, vol. 38, no. 2, 2003.

[2] M. Sivilotti, *Wiring considerations in analog VLSI systems, with application to field-programmable networks*. PhD thesis, California Institute of Technology, Pasadena, CA, 1991.

[3] M. Mahowald, *An analog VLSI system for stereoscopic vision.* Boston, MA: Kluwer Academic Publishers, 1994.

[4] J. Lazzaro, J. Wawrzynek, M. Mahowald, M. Sivilotti, and D. Gillespie, "Silicon auditory processors as computer peripherals," *IEEE Trans. Neural Networks*, vol. 4, no. 3, pp. 523–528, 1993.

[5] K. A. Boahen, "Point-to-point connectivity between neuromorphic chips using address events," *IEEE Trans. Circuits & Systems II*, vol. 47, no. 5, pp. 416–434, 2000.

[6] C. M. Higgins and C. Koch, "Multi-chip neuromorphic motion processing," in *Proc. 20th Anniversary Conference on Advanced Research in VLSI* (D. S. Wills and S. P. DeWeerth, eds.), (Los Alamitos, CA), pp. 309–323, IEEE Computer Society, 1999.

[7] S.-C. Liu, J. Kramer, G. Indiveri, T. Delbrück, and R. Douglas, "Orientation-selective aVLSI spiking neurons," in *Advances in Neural Information Processing Systems 14* (T. G. Dietterich, S. Becker, and Z. Ghahramani, eds.), Cambridge, MA: MIT Press, 2002.

[8] G. Indiveri, A. M. Whatley, and J. Kramer, "A reconfigurable neuromorphic VLSI multi-chip system applied to visual motion computation," in *Proc. MicroNeuro'99*, Apr. 1999.

[9] D. H. Goldberg, G. Cauwenberghs, and A. G. Andreou, "Probabilistic synaptic weighting in a reconfigurable network of VLSI integrate-and-fire neurons," *Neural Networks*, vol. 14, no. 6-7, pp. 781–793, 2001.

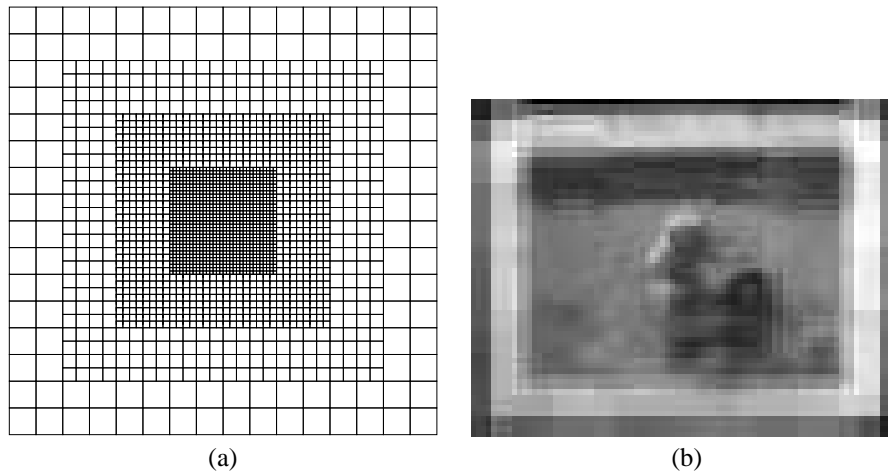

|  |  |
|:---:|:---:|
| (a) | (b) |

Figure 7: (a) Stage 3 foveation network. The $32 \times 32$ pixel high-resolution fovea (center) is surrounded by lower-resolution areas where $2 \times 2$, $4 \times 4$, and $8 \times 8$ blocks of OR neurons (shown as non-overlapping for clarity) project to single IFAT cells. The address space for inputs to the foveation network is $128 \times 128$. [18]. (b) Output of stage 3, as implemented on the IFAT, with the fovea centered on the location with the maximum firing rate in Figure 6b, from stage 2. Peripheral pixels that receive no input are not shown.

[10] S. R. Deiss, R. J. Douglas, and A. M. Whatley, "A pulse-coded communications infrastructure for neuromorphic systems," in *Pulsed Neural Networks* (W. Maass and C. M. Bishop, eds.), pp. 157–178, Cambridge, MA: MIT Press, 1999.

[11] M. Mahowald and R. Douglas, "A silicon neuron," *Nature*, vol. 354, pp. 515–518, 1991.

[12] R. J. Vogelstein, F. Tenore, R. Philipp, M. S. Adlerstein, D. H. Goldberg, and G. Cauwenberghs, "Spike timing-dependent plasticity in the address domain," in *Advances in Neural Information Processing Systems 15* (S. Becker, S. Thrun, and K. Obermayer, eds.), Cambridge, MA: MIT Press, 2003.

[13] R. J. Vogelstein, U. Mallik, and G. Cauwenberghs, "Silicon spike-based synaptic array and address-event transceiver," in *Proc. ISCAS'04*, vol. 5, (Vancouver, BC), pp. 385–388, 2004.

[14] C. Koch, *Biophysics of Computation: Information Processing in Single Neurons*. New York, NY: Oxford University Press, 1999.

[15] C. Koch and S. Ullman, "Shifts in selective visual attention: towards the underlying neural circuitry," *Human Neurobiology*, vol. 4, pp. 219–227, 1985.

[16] L. Itti, E. Niebur, and C. Koch, "A model of saliency-based fast visual attention for rapid scene analysis," *IEEE Trans. Pattern Analysis & Machine Intelligence*, vol. 20, no. 11, pp. 1254–1259, 1998.

[17] T. Horiuchi, T. Morris, C. Koch, and S. P. DeWeerth, "Analog VLSI circuits for attention-based, visual tracking," in *Advances in Neural Information Processing Systems 9*, pp. 706–712, Cambridge, MA: MIT Press, 1997.

[18] R. J. Vogelstein, U. Mallik, E. Culurciello, G. Cauwenberghs, and R. Etienne-Cummings, "Spatial acuity modulation of an address-event imager," in *ICECS'04*, 2004.

[19] R. J. Vogelstein, U. Mallik, and G. Cauwenberghs, "Reconfigurable silicon array of spiking neurons," *IEEE Trans. Neural Networks*, 2005. (Submitted).

[20] E. Culurciello, R. Etienne-Cummings, and K. Boahen, "Second generation of high dynamic range, arbitrated digital imager," in *Proc. ISCAS'04*, vol. 4, (Vancouver, BC), pp. 828–831, 2004.
